# An exploration-exploitation model based on norepinepherine and dopamine activity

**Samuel M. McClure[*], Mark S. Gilzenrat, and Jonathan D. Cohen**
Center for the Study of Brain, Mind, and Behavior
Princeton University
Princeton, NJ 08544
*smcclure@princeton.edu; mgilzen@princeton.edu; jdc@princeton.edu*

## Abstract

We propose a model by which dopamine (DA) and norepinepherine (NE) combine to alternate behavior between relatively exploratory and exploitative modes. The model is developed for a target detection task for which there is extant single neuron recording data available from locus coeruleus (LC) NE neurons. An exploration-exploitation trade-off is elicited by regularly switching which of the two stimuli are rewarded. DA functions within the model to change synaptic weights according to a reinforcement learning algorithm. Exploration is mediated by the state of LC firing, with higher tonic and lower phasic activity producing greater response variability. The opposite state of LC function, with lower baseline firing rate and greater phasic responses, favors exploitative behavior. Changes in LC firing mode result from combined measures of response conflict and reward rate, where response conflict is monitored using models of anterior cingulate cortex (ACC). Increased long-term response conflict and decreased reward rate, which occurs following reward contingency switch, favors the higher tonic state of LC function and NE release. This increases exploration, and facilitates discovery of the new target.

## 1  Introduction

A central problem in reinforcement learning is determining how to adaptively move between exploitative and exploratory behaviors in changing environments. We propose a set of neurophysiologic mechanisms whose interaction may mediate this behavioral shift. Empirical work on the midbrain dopamine (DA) system has suggested that this system is particularly well suited for guiding exploitative behaviors. This hypothesis has been reified by a number of studies showing that a temporal difference (TD) learning algorithm accounts for activity in these neurons in a wide variety of behavioral tasks [1,2]. DA release is believed to encode a reward prediction error signal that acts to change synaptic weights relevant for producing behaviors [3]. Through learning, this allows neural pathways to predict future expected reward through the relative strength of their synaptic connections

[1]. Decision-making procedures based on these value estimates are necessarily greedy. Including reward bonuses for exploratory choices supports non-greedy actions [4] and accounts for additional data derived from DA neurons [5]. We show that combining a DA learning algorithm with models of response conflict detection [6] and NE function [7] produces an effective annealing procedure for alternating between exploration and exploitation.

NE neurons within the LC alternate between two firing modes [8]. In the first mode, known as the phasic mode, NE neurons fire at a low baseline rate but have relatively robust phasic responses to behaviorally salient stimuli. The second mode, called the tonic mode, is associated with a higher baseline firing and absent or attenuated phasic responses. The effects of NE on efferent areas are modulatory in nature, and are well captured as a change in the gain of efferent inputs so that neuronal responses are potentiated in the presence of NE [9]. Thus, in phasic mode, the LC provides transient facilitation in processing, time-locked to the presence of behaviorally salient information in motor or decision areas. Conversely, in tonic mode, higher overall LC discharge rate increases gain generally and hence increases the probability of arbitrary responding. Consistent with this account, for periods when NE neurons are in the phasic mode, monkey performance is nearly perfect. However, when NE neurons are in the tonic mode, performance is more erratic, with increased response times and error rate [8]. These findings have led to a recent characterization of the LC as a dynamic temporal filter, adjusting the system's relative responsivity to salient and irrelevant information [8]. In this way, the LC is ideally positioned to mediate the shift between exploitative and exploratory behavior.

The parameters that underlie changes in LC firing mode remain largely unexplored. Based on data from a target detection task by Aston-Jones and colleagues [10], we propose that LC firing mode is determined in part by measures of response conflict and reward rate as calculated by the ACC and OFC, respectively [8]. Together, the ACC and OFC are the principle sources of cortical input to the LC [8]. Activity in the ACC is known, largely through human neuroimaging experiments, to change in accord with response conflict [6]. In brief, relatively equal activity in competing behavioral responses (reflecting uncertainty) produces high conflict. Low conflict results when one behavioral response predominates. We propose that increased long-term response conflict biases the LC towards a tonic firing mode. Increased conflict necessarily follows changes in reward contingency. As the previously rewarded target no longer produces reward, there will be a relative increase in response ambiguity and hence conflict. This relationship between conflict and LC firing is analogous to other modeling work [11], which proposes that increased tonic firing reflects increased environmental uncertainty.

As a final component to our model, we hypothesize that the OFC maintains an ongoing estimate in reward rate, and that this estimate of reward rate also influences LC firing mode. As reward rate increases, we assume that the OFC tends to bias the LC in favor of phasic firing to target stimuli.

We have aimed to fix model parameters based on previous work using simpler networks. We use parameters derived primarily from a previous model of the LC by Gilzenrat and colleagues [7]. Integration of response conflict by the ACC and its influence on LC firing was borrowed from unpublished work by Gilzenrat and colleagues in which they fit human behavioral data in a diminishing utilities task. Given this approach, we interpret our observed improvement in model performance with combined NE and DA function as validation of a mechanism for automatically switching between exploitative and exploratory action selection.

## 2 Go-No-Go Task and Core Model

We have modeled an experiment in which monkeys performed a target detection task [10]. In the task, monkeys were shown either a vertical bar or a horizontal bar and were required to make or omit a motor response appropriately. Initially, the vertical bar was the target stimulus and correctly responding was rewarded with a squirt of fruit juice ($r$=1 in the model). Responding to the non-target horizontal stimulus resulted in time out punishment ($r$=-.1; Figure 1A). No responses to either the target or non-target gave zero reward.

After the monkeys had fully acquired the task, the experimenters periodically switched the reward contingency such that the previously rewarded stimulus (target) became the distractor, and vice versa. Following such reversals, LC neurons were observed to change from emitting phasic bursts of firing to the target, to tonic firing following the switch, and slowly back to phasic firing for the new target as the new response criteria was obtained [10].

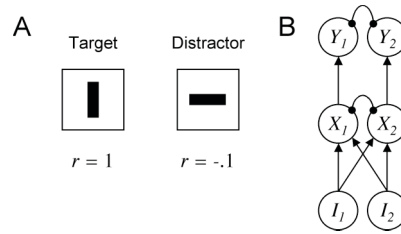

Figure 1: Task and model design. (A) Responses were required for targets in order to obtain reward. Responses to distractors resulted in a minor punishment. No responses gave zero reward. (B) In the model, vertical and horizontal bar inputs ($I_1$ and $I_2$) fed to integrator neurons ($X_1$ and $X_2$) which then drove response units ($Y_1$ and $Y_2$). Responses were made if $Y_1$ or $Y_2$ crossed a threshold while input units were active.

We have previously modeled this task [7,12] with a three-layer connectionist network in which two input units, $I_1$ and $I_2$, corresponding to the vertical and horizontal bars, drive two mutually inhibitory integrator units, $X_1$ and $X_2$. The integrator units subsequently feed two response units, $Y_1$ and $Y_2$ (Figure 1B). Responses are made whenever output from $Y_1$ or $Y_2$ crosses a threshold level of activity, $\theta$. Relatively weak cross connections from each input unit to the opposite integrator unit ($I_1$ to $X_2$ and $I_2$ to $X_1$) are intended to model stimulus similarity.

Both the integrator and response units were modeled as noisy, leaky accumulators:

$$\dot{X}_i = -X_i + w_{X_i I_i} I_i + w_{X_i I_j} I_j - w_{X_i X_j} f(X_j) + \xi_i \qquad (1)$$

$$\dot{Y}_i = -Y_i + w_{Y_i X_i} f(X_i) - w_{Y_i Y_j} f(Y_j) + \xi_i. \qquad (2)$$

The $\xi_i$ terms represent stochastic noise variables. The response function for each unit is sigmoid with gain, $g_t$, determined by current LC activity (Eq. 9, below)

$$f(X) = \left(1 + e^{-g_t(X-b)}\right)^{-1}. \qquad (3)$$

Response units, $Y$, were given a positive bias, $b$, and integrator units were unbiased. All weight values, biases, and variance of noise are as reported in [7].

Integration was done with a Euler method at time steps of 0.02. Simulation of stimulus presentations involved setting one of the input units to a value of 1.0 for 20 units of model time. Activation of $I_1$ and $I_2$ were alternated and 20 units of model time were allowed between presentations for the integrator and response units to relax to baseline levels of activity. Input 1 was initially set to be the target and input 2 the distractor. After 50 presentations of $I_1$ and $I_2$ the reward contingencies were switched; the model was run through 6 such blocks and reversals. The response during each stimulus presentation was determined by which of the two response units first crossed a threshold of output activity (i.e. $f(Y_1) > \theta$), or was a no response if neither unit crossed threshold.

# 3  Performance of model with DA-mediated learning

In order to obtain a benchmark level of performance to compare against, we first determined how learning progresses with DA-mediated reinforcement learning alone. A reward unit, $r$, was included that had activity 0 except at the end of each stimulus presentation when its activity was set equal to the obtained reward outcome. Inhibitory inputs from the response units served as measures of expected reward. At the end of every trial, the DA unit, $\delta$, obtained a value given by

$$\delta(t) = r(t) - w_{\delta Y_1} Z(Y_1(t)) - w_{\delta Y_2} Z(Y_2(t)) \tag{4}$$

where $Z(Y)$ is a threshold function that is 1 if $f(Y) \geq \theta$ and is 0 otherwise.

The output of dopamine neurons was used to update the weights along the pathway that lead to the response. Thus, at the end of every stimulus presentation, the weights between response units and DA neurons were updated according to

$$w_{\delta Y_i}(t+1) = w_{\delta Y_i}(t) + \lambda \delta(t) Z(Y_i) \tag{5}$$

where the learning rate, $\lambda$, was set to 0.3 for all simulations. This learning rule allowed the weights to converge to the expected reward for selecting each of the two actions. Weights between integrator and response units were updated using the same rule as in Eq. 5, except the weights were restricted to a minimum value of 0.8. When the weight values were allowed to decrease below 0.8, sufficient activity never accumulated in the response units to allow discovery to new reward contingencies.

As the model learned, the weights along the target pathway obtained a maximum value while those along the distractor pathway obtained a minimum value. After reversals, the model initially adapted by reducing the weights along the pathway associated with the previous target. The only way the model was able to obtain the new target was by noise pushing the new target response unit above threshold. Because of this, the performance of the model was greatly dependent of the value of the threshold used in the simulation (Figure 2B). When the threshold was low relative to noise, the model was able to quickly adapt to reversals. However, this also resulted in a high rate of responding to non-target stimuli even after learning. In order to reduce responding to the distractor, the threshold had to be raised, which also increased the time required to adapt following reward reversals.

The network was initialized with equal preference for responding to input 1 or 2, and generally acquired the initial target faster than after reversals (see Figure 2B). Because of this, all subsequent analyses ignore this first learning period. For each value of threshold studied, we ran the model 100 times. Plots shown in Figures 2 and 3 show the probability that the model responded, when each input was activated, as a function of trial number (i.e. $P(f(Y_i) \geq \theta \mid I_i = 1)$).

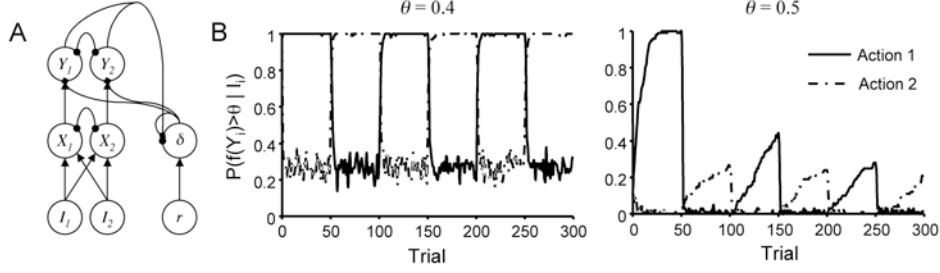

Figure 2: Model performance with DA alone. (A) DA neurons, $\delta$, modulated weights from integrator to response units in order to modulate the probability of responding to each input. (B) The model successfully increases and decreases responding to inputs 1 and 2 as reward contingencies reverse. However, the model is unable to simultaneously obtain the new response quickly and maintain a low error rate once the response is learned. When threshold is relatively low (left plot), the model adapts quickly but makes frequent responses to the distractor. At higher threshold, responses are correctly omitted to the distractor, but the model acquires the new response slowly.

## 4   Improvement with NE-mediated annealing

We used the FitzHugh-Nagumo set of differential equations to model LC activity. (These equations are generally used to model individual neurons, but we use them to model the activity in the nucleus as a whole.) Previous work has shown that these equations, with simple modifications, capture the fundamental aspects of tonic and phasic mode activity in the LC [7]. The FitzHugh-Nagumo equations involve two interacting variables $v$ and $u$, where $v$ is an activity term and $u$ is an inhibitory dampening term. The output of the LC is given by the value of $u$, which conveniently captures the fact that the LC is self-inhibitory and that the post-synaptic effect of NE release is somewhat delayed [7].

The model included two inputs to the LC from the integrator units ($X_1$ and $X_2$) with modifiable weights. The state of the LC is then given by

$$\tau_v \dot{v} = v(\alpha - v)(v - 1) - u + w_{vX_1} f(X_1) + w_{vX_2} f(X_2) \tag{6}$$

$$\tau_u \dot{u} = h(v) - u \tag{7}$$

where the function $h$ is defined by

$$h(v) = Cv + (1 - C)d \tag{8}$$

and governs the firing mode of the LC. In order to change firing mode, $h$ can be modified so that the dynamics of $u$ depend entirely on the state of the LC or so that the dynamics are independent of state. This alternation is governed by the parameter $C$. When $C$ is equal to 1.0, the model is appropriately dampened and can burst sharply and return to a relatively low baseline level of activity (phasic mode). When $C$ is small, the LC receives a fixed level of inhibition, which simultaneously reduces bursting activity and increases baseline activity (tonic mode) [7].

The primary function of the LC in the model is to modify the gain, $g$, of the response function of the integrator and response units as in equation 3. We let gain be a linear function of $u$ with base value $G$ and dependency on $u$ given by $k$

$$g_t = G + ku_t. \tag{9}$$

The value of $C$ was updated after every trial by measures of response conflict and reward rate. Response conflict was calculated as a normalized measure of the energy in the response units during the trial. For convenience, define $\mathbf{Y_1}$ to be a vector of the activity in unit $Y_1$ at each point of time during a trial, $f(Y_1(t))$. Let $\mathbf{Y_2}$ be defined similarly. The conflict during the trial is

$$K = \frac{\mathbf{Y_1} \cdot \mathbf{Y_2}}{|\mathbf{Y_1}||\mathbf{Y_2}|} \tag{10}$$

which correctly measures energy since $Y_1$ and $Y_2$ are connected with weight $-1$. This normalization procedure was necessary to account for changes in the magnitude of $Y_1$ and $Y_2$ activity due to learning.

Based on previous work [8], we let conflict modify $C$ separately based on a short-term, $K_S$, and long-term, $K_L$, measure. The variable $K_S$ was updated at the end of every $T$th trial according to

$$K_S(T+1) = (1 - \varepsilon_S)K_S(T) + \varepsilon_S K(T). \tag{11}$$

where $\varepsilon_S$ was 0.2 and $K_S(T+1)$ was used to calculate the value of $C$ used for the $T+1$th trial. $K_L$ was update with the same rule as $K_S$ except $\varepsilon_L$ was 0.05. We let short- and long-term conflict have opposing effect on the firing mode of the LC. This was developed previously to capture human behavior in a diminishing utilities task. When short-term conflict increases, the LC is biased towards phasic firing (increased $C$). This allows the model to recover from occasional errors. However, when long-term conflict increases this is taken to indicate that the current decision strategy is not working. Therefore, increased long-term conflict biases the LC to the tonic mode so as to increase response volatility.

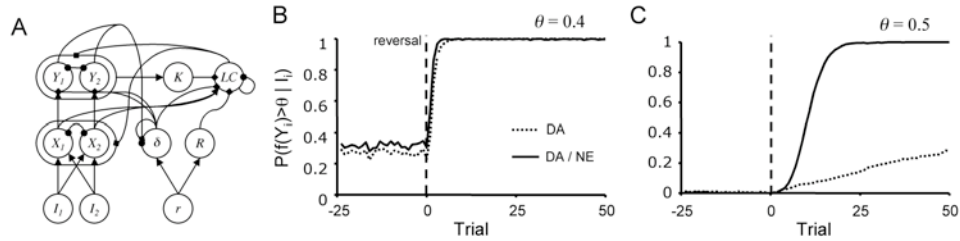

Figure 3: Model performance with DA and NE. (A) The full model includes a conflict detection unit, $K$, and a reward rate measure, $R$, which combine to modify activity in the $LC$. The $LC$ modifies the gain in the integrator and response units. (B) The benefit of including the LC in the model is insignificant when the response threshold is regularly crossed by noise alone, and hence when the error rate is high. (C) However, when the threshold is greater and error rate lower, NE dramatically improves the rate at which the new reward contingencies are learned after reversal.

Reward rate, $R$, was updated at the end of every trial according to

$$R(T+1) = (1 - \varepsilon_R)R(T) + \varepsilon_R r \tag{12}$$

where $r$ is the reward earned on the $T$th trial. Increased reward rate was assumed to bias the LC to phasic firing.

Reward rate, short-term conflict, and long-term conflict updated $C$ according to

$$C = \sigma(K_S)\big(1 - \sigma(K_L)\big)\sigma(R) \tag{13}$$

where each σ is a sigmoid function with a gain of 6.0 and no bias as determined by fitting to behavior with previous models.

As with the model with DA alone, the effect of NE depended significantly on the value of the threshold $\theta$. When $\theta$ was small, the improvement afforded by the LC was negligible (Figure 3B). However, when the threshold was significantly greater than noise, the improvement was substantial (Figure 3C).

Monkeys were able to perform this task with accuracy greater than 90% and simultaneously were able to adapt to reversals within 50 trials [10]. While it is impossible to compare the output of our model with monkey behavior, we can make the qualitative assertion that, as with monkeys, our NE-based annealing model allows for high accuracy (and high threshold) decision-making while preserving adaptability to changes in reward contingencies. In order to better demonstrate this improvement, we fit single exponential curves to the plots of probability of accurately responding to the new target by trial number (as in Figure 3B,C). Shown in Figure 4 is the time constant for these exponential fits, which we term the discovery time constant, for different values of the threshold. As can be seen, the model with NE-mediated annealing maintains a relatively fast discovery time even as the threshold becomes relatively large.

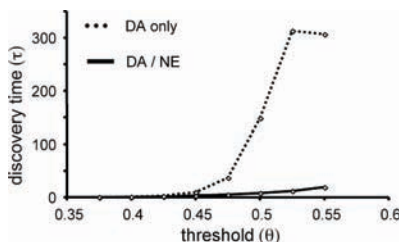

Figure 4: Summary of model performance with and without NE.

## 5  Discussion

We have demonstrated that a model incorporating behavioral and learning effects previously ascribed to DA and NE produces an adaptive mechanism for switching between exploratory and exploitative decision-making. Our model uses measures of response conflict and reward rate to modify LC firing mode, and hence to change network dynamics in favor of more or less volatile behavior. In essence, combining previous models of DA and NE function produces a performance-based auto-annealing algorithm.

There are several limitations to this model that can be remedied by greater sophistication in the learning algorithm. The primary limitation is that the model varies between more or less volatile action selection only over the range of reward relevant to our studied task. Model parameters could be altered on a task-by-task basis to correct this; however, a more general scheme may be accomplished with a mean reward learning algorithm [13]. It has previously been argued that DA neurons may actually emit an average reward TD error [14]. This change may require allowing both short- and long-term reward rate control the LC firing mode (Eq. 13).

Another limitation of this model is that, while exploration is increased as performance measures wane, exploration is not managed intelligently. This does not significantly affect the performance of our model since there are only two available actions. As the number of alternatives increases, rapid learning may require something akin to reward bonuses [4,5].

Understanding the interplay between DA and NE function in learning and decision-making is also relevant for understanding disease. Numerous psychiatric disorders are known to involve dysregulation of NE and DA release. Furthermore, hallmark features of ADHD and schizophrenia include cognitive disorders in which behavior appears either too volatile (ADHD) or too inflexible (schizophrenia) [15,16]. Improved models of DA-NE interplay during learning and decision-making, coupled with empirical data, may simultaneously improve knowledge of how the brain handles the exploration-exploitation dilemma and how this goes awry in disease.

## Acknowledgments

This work was supported by NIH grants P50 MH62196 and MH065214.

## References

[1] Montague, P.R. Dayan, P., Sejnowski, T.J. (1996) A framework for mesencephalic dopamine systems based on predictive Hebbian learning. *J. Neurosci.* **16**: 1936-1947.

[2] Schultz, W. Dayan, P. & Montague, P.R. (1997) A neural substrate for prediction and reward. *Science* **275**: 1593-1599.

[3] Reynolds, J.N., Hyland, B.I., Wickens, J.R. (2001) A cellular mechanism of reward-related learning. *Nature* **413**: 67-70.

[4] Sutton, R.S. (1990) Integrated architectures for learning, planning, and reacting based on approximated dynamic programming. *Mach. Learn., Proc. 7th International Conf.* 216-224.

[5] Kakade, S., Dayan, P. (2002) Dopamine: generalization and bonuses. *Neural Networks* **15**: 549-559.

[6] Botvinick, M.M., Braver, T.S., Barch, D.M., Carter, C.S., Cohen, J.D. (2001) Conflict monitoring and cognitive control. *Psychol. Rev.* **108**: 624-652.

[7] Gilzenrat, M.S., Holmes, B.D., Rajkowski, J., Aston-Jones, G., Cohen, J.D. (2002) Simplified dynamics in a model of noradrenergic modulation of cognitive performance. *Neural Networks* **15**: 647-663.

[8] Aston-Jones, G., Cohen, J.D. (2005) An integrative theory of locus coeruleus-norepinepherine function. *Ann. Rev. Neurosci.* **28**: 403-450.

[9] Servan-Schreiber, D., Printz, H., Cohen, J.D. (1990) A network model of catecholamine effects: gain, signal-to-noise ratio and behavior. *Science* **249**: 892-895.

[10] Aston-Jones, G., Rajkowski, J., Kubiak, P. (1997) Conditioned responses of monkey locus coeruleus neurons anticipate acquisition of discriminative behavior in a vigilance task. *Neuroscience* **80**: 697-715.

[11] Yu, A., Dayan, P. (2005) Uncertainty, neuromodulation and attention. *Neuron* **46**: 681-92.

[11] Usher, M., Cohen, J.D., Rajkowski, J., Aston-Jones, G. (1999) The role of the locus coeruleus in the regulation of cognitive performance. *Science* **283**: 549-554.

[12] Schwartz, A. (1993) A reinforcement learning method for maximizing undiscounted rewards. In: *Proc. 10th International Conf. Mach. Learn.* (pp. 298-305). San Mateo, CA: Morgan Kaufmann.

[13] Daw, N.D., Touretzky, D.S. (2002) Long-term reward prediction in TD models of the dopamine system. *Neural Computation* **14**: 2567-2583.

[14] Goldberg, T.E., Weinberger, D.R., Berman, K.F., Pliskin, N.H., Podd, M.H. (1987) Further evidence for dementia of the prefrontal type in schizophrenia? A controlled study teaching the Wisconsin Card Sorting Test. *Arch. Gen. Psychiatry* **44**: 1008-1014.

[15] Barkley, R.A. (1997) Behavioural inhibition, sustained attention, and executive functions: constructing a unified theory of AD/HD. *Psychol. Bull.* **121**: 65-94.
